# Active inference in concept learning

**Jonathan D. Nelson**
Department of Cognitive Science
University of California, San Diego
La Jolla, CA 92093-0515
*jnelson@cogsci.ucsd.edu*

**Javier R. Movellan**
Department of Cognitive Science
University of California, San Diego
La Jolla, CA 92093-0515
*movellan@inc.ucsd.edu*

## Abstract

People are active experimenters, not just passive observers, constantly seeking new information relevant to their goals. A reasonable approach to active information gathering is to ask questions and conduct experiments that maximize the expected information gain, given current beliefs (Fedorov 1972, MacKay 1992, Oaksford & Chater 1994). In this paper we present results on an exploratory experiment designed to study people's active information gathering behavior on a concept learning task (Tenenbaum 2000). The results of the experiment are analyzed in terms of the expected information gain of the questions asked by subjects.

In scientific inquiry and in everyday life, people seek out information relevant to perceptual and cognitive tasks. Scientists perform experiments to uncover causal relationships; people saccade to informative areas of visual scenes, turn their head towards surprising sounds, and ask questions to understand the meaning of concepts.

Consider a person learning a foreign language, who notices that a particular word, "tikos," is used for baby moose, baby penguins, and baby cheetahs. Based on those examples, he or she may attempt to discover what tikos really means. Logically, there are an infinite number of possibilities. For instance, tikos could mean *baby animals*, or simply *animals*, or even *baby animals and antique telephones*. Yet a few examples are enough for human learners to form strong intuitions about what meanings are most likely.

Suppose you can point to a baby duck, an adult duck, or an antique telephone, to inquire whether that object is "tikos." Your goal is to figure out what "tikos" means. Which question would you ask? Why? When the goal is to learn as much as possible about a set of concepts, a reasonable strategy is to choose those questions which maximize the expected information gain, given current beliefs (Fedorov 1972, MacKay 1992, Oaksford & Chater 1994). In this paper we present preliminary results on an experiment designed to quantify the information value of the questions asked by subjects on a concept learning task.

## 1.1 Tenenbaum's number concept task

Tenenbaum (2000) developed a Bayesian model of number concept learning. The model describes the intuitive beliefs shared by humans about simple number

concepts, and how those beliefs change as new information is obtained, in terms of subjective probabilities. Suppose a subject has been told that the number 16 is consistent with some unknown number concept. With its current parameters, the model predicts that the subjective probability that the number 8 will also be consistent with that concept is about 0.35. Tenenbaum (2000) included both *mathematical* and *interval* concepts in his number concept space. Interval concepts were sets of *numbers between n and m*, where $1 \leq n \leq 100$, and $n \leq m \leq 100$, such as *numbers between 5 and 8*, and *numbers between 10 and 35*. There were 33 mathematical concepts: *odd numbers, even numbers, square numbers, cube numbers, prime numbers, multiples of n* ($3 \leq n \leq 12$), *powers of n* ($2 \leq n \leq 10$), and *numbers ending in n* ($1 \leq n \leq 9$). Tenenbaum conducted a number concept learning experiment with 8 subjects and found a correlation of 0.99 between the average probability judgments made by subjects and the model predictions. To evaluate how well Tenenbaum's model described our population of subjects, we replicated his study, with 81 subjects. We obtained a correlation of .87 between model predictions and average subject responses. Based on these results we decided to extend Tenenbaum's experiment, and allow subjects to actively ask questions about number concepts, instead of just observing examples given to them. We used Tenenbaum's model to obtain estimates of the subjective probabilities of the different concepts given the examples at hand. We hypothesized that the questions asked by subjects would have high information value, when information value was calculated according to the probability estimates produced by Tenenbaum's model.

## 1.2   Infomax sampling

Consider the following problem. A subject is given examples of numbers that are consistent with a particular concept, but is not told the concept itself. Then the subject is allowed to pick a number, to test whether it follows the same concept as the examples given. For example, the subject may be given the numbers 2, 6 and 4 as examples of the underlying concept and she may then choose to ask whether the number 8 is also consistent with the concept. Her goal is to guess the correct concept.

We formalize the problem using standard probabilistic notation: random variables are represented with capital letters and specific values taken by those variables are represented with small letters. The random variable $C$ represents the correct concept on a given trial. Notation of the form "$C=c$" is shorthand for the event that the random variable $C$ takes the specific value c. We represent the examples given to the subjects by the random vector $X$. The subject beliefs about which concepts are probable prior to the presentation of any examples is represented by the probability function $P(C = c)$. The subject beliefs after the examples are presented is represented by $P(C = c \mid X = x)$. For example, if $c$ is the concept *even numbers* and $x$ the numbers "2, 6, 4", then $P(C = c \mid X = x)$ represents subjects' posterior probability that the correct concept is *even numbers*, given that 2, 6, and 4 are positive examples of that concept. The binary random variable $Y_n$ represents whether the number $n$ is a member of the correct concept. For example, $Y_8 = 1$ represents the event that 8 is an element of the correct concept, and $Y_8 = 0$ the event that 8 is not. In our experiment subjects are allowed to ask a question of the form "*is the number n an element of the concept?*". This is equivalent to finding the value taken by the random variable $Y_n$, in our formalism.

We evaluate how good a question is in terms of the information about the correct concept expected for that question, given the example vector $X=x$. The expected information gain for the question *"Is the number n an element of the concept?"* is given by the following formula:

$$I(C, Y_n \mid X = x) = H(C \mid X = x) - H(C \mid Y_n, X = x),$$

where $H(C \mid X = x)$ is the uncertainty (entropy) about of the concept $C$ given the example numbers in $x$

$$H(C \mid X = x) = -\sum_c P(C = c \mid X = x) \, \log_2 \, P(C = c \mid X = x),$$

and

$$H(C \mid Y_n, X = x) =$$

$$-\sum_{c \in C} P(C = c \mid X = x) \sum_{v=0}^{1} P(Y_n = v \mid C = c, X = x) \, \log_2 \, P(C = c \mid Y_n = v, X = x),$$

is the uncertainty about $C$ given the active question $Y_n$ and the example vector $x$. We consider only binary questions, of the form "is $n$ consistent with the concept?" so the maximum information value of any question in our experiment is one bit. Note how information gain is relative to a probability model $P$ of the subjects' internal beliefs. Here we approximate these subjective probabilities using Tenenbaum's (2000) number concept model.

An information-maximizing strategy (infomax) prescribes asking the question with the highest expected information gain, e.g., the question that minimizes the expected entropy, over all concepts. Another strategy of interest is *confirmatory sampling*, which consists of asking questions whose answers are most likely to confirm current beliefs. In other domains it has been proposed that subjects have a bias to use confirmatory strategies regardless of their information value (Klayman & Ha 1987, Popper 1959, Wason 1960). Thus, it is interesting to see whether people use a confirmatory strategy on our concept learning task and to evaluate how informative such a strategy would be.

## 2   Human sampling in the number concept game

Twenty-nine undergraduate students, recruited from Cognitive Science Department classes at the University of California, San Diego, participated in the experiment.[1] Subjects gave informed consent, and received either partial course credit for required study participation, or extra course credit, for their participation. The experiment began with the following instructions:

*Often it is possible to have a good idea about the state of the world, without completely knowing it. People often learn from examples, and this study explores how people do so. In this experiment, you will be given examples of a hidden number rule. These examples will be randomly chosen from the numbers between 1 and 100 that follow the rule. The true rule will remain hidden, however. Then you will be able to test an additional number, to see if it follows that same hidden rule. Finally, you will be asked to give your best estimation of what the true hidden rule is, and the chances that you are right. For instance, if the true hidden rule were "multiples of 11", you might see the examples 22 and 66. If you thought the rule were" multiples of 11", but also possibly "even numbers", you could test a number of your choice, between 1-100, to see if it also follows the rule.*

On each trial subjects first saw a set of examples from the correct concept. For instance, if the concept were *even numbers*, subjects might see the numbers "2, 6, 4" as examples. Subjects were then given the opportunity to test a number of their choice. Subjects were given feedback on whether the number they tested was an element of the correct concept.

We wrote a computer program that uses the probability estimates provided by Tenenbaum (2000) model to compute the information value of any possible question in the number concept task. We used this program to evaluate the information value of the questions asked by subjects, the questions asked by an infomax strategy, the questions asked by a confirmatory strategy, and the questions asked by a random sampling strategy. The infomax strategy was determined as described above. The random strategy consisted of randomly testing a number between 1 and 100 with equal probability. The confirmatory strategy consisted of testing the number (excluding the examples) that had the highest posterior probability, as given by Tenenbaum's model, of being consistent with the correct concept.

# 3 Results

Results for nine representative trials are discussed. The trials are grouped into three types, according to the posterior beliefs of Tenenbaum's model, after the example numbers have been seen. The average information value of subjects' questions, and of each simulated sampling strategy, are given in Table 1. The specific questions subjects asked are considered in Sections 3.1-3.3.

| *Trial type* | *Single example, high uncertainty* | | | *Multiple example, low uncertainty* | | | *Interval* | | |
|---|---|---|---|---|---|---|---|---|---|
| *Examples* | *16* | *60* | *37* | *16, 8, 2, 64* | *60, 80, 10, 30* | *81, 25, 4, 36* | *16, 23, 19, 20* | *60, 51, 57, 55* | *81, 98, 96, 93* |
| *Subjects* | .70 | .72 | .73 | .00 | .06 | 0.00 | .47 | .37 | .31 |
| *Infomax* | .97 | 1.00 | 1.00 | .01 | .32 | 0.00 | 1.00 | .99 | 1.00 |
| *Confirmation* | .97 | 1.00 | 1.00 | .00 | .00 | 0.00 | 0.00 | 0.00 | 0.00 |
| *Random* | .35 | .54 | .52 | .00 | .04 | 0.00 | .17 | .20 | .14 |

Table 1. Information value, as assessed using the subjective probabilities in Tenenbaum's number concept model, of several sampling strategies. Information value is measured in bits.

## 3.1 Single example, high uncertainty trials

On these trials Tenenbaum's model is relatively uncertain about the correct concepts and gives some probability to many concepts. Interestingly, the confirmatory strategy is identical to the infomax strategy on each of these trials, suggesting that a confirmatory sampling strategy may be optimal under conditions of high uncertainty. Consider the trial with the example number 16. On this trial, the concepts *powers of 4*, *powers of 2*, and *square numbers* each have moderate posterior probability (.28, .14, and .09, respectively).

These trials provided the best qualitative agreement between infomax predictions and subjects' sampling behavior. Unfortunately the results are inconclusive since on these trials both infomax and confirmatory strategy make the same predictions. On the trial with the example number 16, subjects' modal response (8 of 29 subjects)

was to test the number 4. This was actually the most informative question, according to Tenenbaum's model. Several subjects (8 of 29) tested other square numbers, such as 49, 36, or 25, which also have high information value, relative to Tenenbaum's number concept model (Figure 1). Subjects' questions also had a high information value on the trial with the example number 37, and the trial with the example number 60.

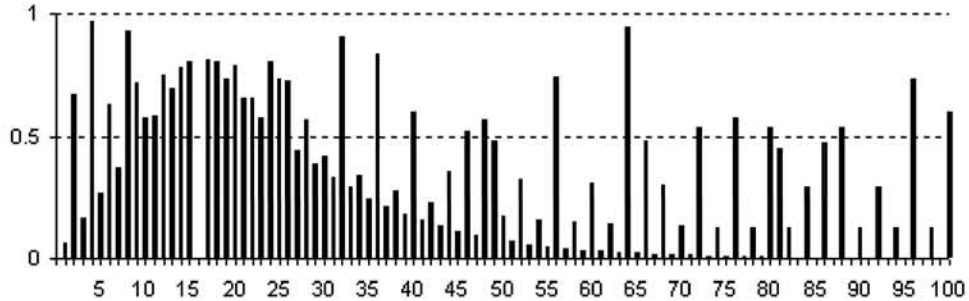

Figure 1. Information value of sampling each number, in bits, given that the number 16 is consistent with the correct concept.

## 3.2 Multiple example, low uncertainty trials

On these trials Tenenbaum's model gives a single concept very high posterior probability. When there is little or no information value in any question, infomax makes no particular predictions regarding which questions are best. Most subjects tested numbers that were consistent with the most likely concept, but not specifically given as examples. This behavior matches the confirmatory strategy.

On the trial with the examples 81, 25, 4, and 36, the model gave probability 1.00 to the concept *square numbers*. On this trial, the most commonly tested numbers were 49 (11 of 29 subjects) and 9 (4 of 29 subjects). No sample is expected to be informative on this trial, because overall uncertainty is so low.

On the trial with the example numbers 60, 80, 10, and 30, the model gave probability .94 to the concept *multiples of 10*, and probability .06 to the concept *multiples of 5*. On this trial, infomax tested odd multiples of 5, such as 15, each of which had expected information gain of 0.3 bits. The confirmatory strategy tested non-example multiples of 10, such as 50, and had an information value of 0 bits. Most subjects (17 of 29) followed the confirmatory strategy; some subjects (5 of 29) followed the infomax strategy.

## 3.3 Interval trials

It is desirable to consider situations in which: (1) the questions asked by the infomax strategy are different than the questions asked by the confirmatory strategy, and (2) the choice of questions matters, because some questions have high information value. Trials in which the correct concept is an interval of numbers provide such situations. Consider the trial with the example numbers 16, 23, 19, and 20. On this trial, and the other "interval" trials, the concept learning model is certain that the correct concept is of the form *numbers between m and n*, because the observed examples rule out all the other concepts. However, the model is not certain of the precise endpoints, such as whether the concept is *numbers between 16 and 23*, or *numbers between 16 and 24*, etc. Infomax tests numbers near to, but outside of, the range spanned by the examples, such as 14 or 26, in this example (See Figure 2 at left).

What do subjects do? Two patterns of behavior, each observed on all three interval trials, can be distinguished. The first is to test numbers outside of, but near to, the range of observed examples. On the trial with example numbers between 16 and 23, a total of 15 of 29 subjects tested numbers between 10-15, or 24-30. This behavior is qualitatively similar to infomax.

The second pattern of behavior, which is shown by about one third of the subjects, consists of testing (non-example) numbers within the range spanned by the observed examples. If one is certain that the concept at hand is an interval then asking about numbers within the range spanned by the observed examples provides no information (Figure 2 at left). Yet some subjects consistently ask about these numbers. Based on this surprising result, we went back to the results of Experiment 1, and reanalyzed the accuracy of Tenenbaum's model on trials in which the model gave high probability to interval concepts. We found that on such trials the model significantly deviated from the subjects' beliefs. In particular, subjects gave probability lower than one that non-example numbers within the range spanned by observed examples were consistent with the true concept. The model, however, gives all numbers within the range spanned by the examples probability 1. See Figure 2, at right, and note the difference between subjective probabilities (points) and the model's estimate of these probabilities (solid line). We hypothesize that the apparent uninformativeness of the questions asked by subjects on these trials is due to imperfections in the current version of Tenenbaum's model, and are working to improve the model's descriptive accuracy, to test this hypothesis.

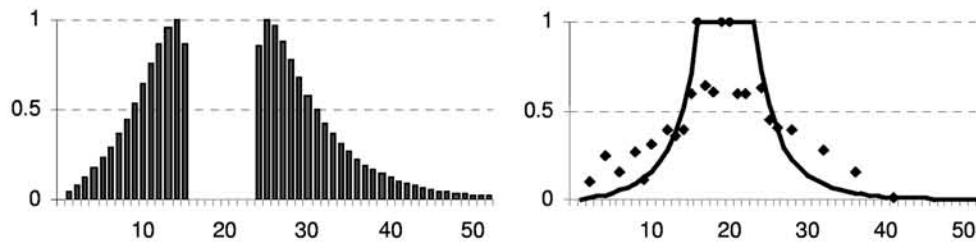

Figure 2. Information value, relative to Tenenbaum's model, of sampling each number, given the example numbers 16, 23, 19, and 20 (left). In this case the model is certain that the correct concept is some interval of numbers; thus, it is not informative to ask questions about numbers within the range spanned by that examples. At right, the probability that each number is consistent with the correct concept. Our subjects' mean probability rating is denoted with points (n = 81, from our first experiment). Tenenbaum's model's approximation of these probabilities is denoted with the solid line.

## 4  Discussion

This paper presents exploratory work in progress that attempts to analyze active inference from the point of view of the rational approach to cognition (Anderson, 1990; Oaksford and Chater, 1994).

First we performed a large scale replication of Tenenbaum's number concept experiment (Tenenbaum, 2000), in which subjects estimated the probability that each of several test numbers were consistent with the same concept as some example numbers. We found a correlation of 0.87 between our subjects' average probability estimates and the probabilities predicted by Tenenbaum's model. We then extended Tenenbaum's experiment by allowing subjects to ask questions about the concepts at hand. Our goal was to evaluate the information value of the

questions asked by subjects. We found that in some situations, a simple confirmatory strategy maximizes information gain. We also found that the current version of Tenenbaum's number concept model has significant imperfections, which limit its ability to estimate the informativeness of subjects' questions. We expect that modifications to Tenenbaum's model will enable infomax to predict sampling behavior in the number concept domain. We are performing simulations to explore this point. We are also working to generalize the infomax analysis of active inference to more complex and natural problems.

## Acknowledgments

We thank Josh Tenenbaum, Gedeon Déak, Jeff Elman, Iris Ginzburg, Craig McKenzie, and Terry Sejnowski for their ideas; and Kent Wu and Dan Bauer for their help in this research. The first author was partially supported by a Pew graduate fellowship during this research.

## Footnotes

[1] Full stimuli are posted at http://hci.ucsd.edu/~jnelson/pages/study.html

## References

Anderson, J. R. (1990). *The adaptive character of thought.* New Jersey: Erlbaum.

Fedorov, V. V. (1972). *Theory of optimal experiments.* New York: Academic Press.

Klayman, J.; Ha, Y. (1987). Confirmation, disconfirmation, and information in hypothesis testing. *Psychological Review*, 94, 211-228.

MacKay, D. J. C. (1992). Information-based objective functions for active data selection. *Neural Computation*, 4, 590-604.

Oaksford, M.; Chater, N. (1994). A rational analysis of the selection task as optimal data selection. *Psychological Review*, 101, 608-631.

Popper, K. R. (1959). *The logic of scientific discovery.* London: Hutchnison.

Tenenbaum, J. B. (2000). Rules and similarity in concept learning. In *Advances in Neural Information Processing Systems*, 12, Solla, S. A., Leen, T. K., Mueller, K.-R. (eds.), 59-65.

Wason, P. C. (1960). On the failure to eliminate hypotheses in a conceptual task. *Quarterly Journal of Experimental Psychology.* 12, 129-140.
